# Solving Decision Problems with Limited Information

**Denis D. Mauá**
IDSIA
Manno, CH 6928
denis@idsia.ch

**Cassio P. de Campos**
IDSIA
Manno, CH 6928
cassio@idsia.ch

## Abstract

We present a new algorithm for exactly solving decision-making problems represented as an influence diagram. We do not require the usual assumptions of no forgetting and regularity, which allows us to solve problems with limited information. The algorithm, which implements a sophisticated variable elimination procedure, is empirically shown to outperform a state-of-the-art algorithm in randomly generated problems of up to 150 variables and $10^{64}$ strategies.

## 1 Introduction

In many tasks, bounded resources and physical constraints force decisions to be made based on limited information [1, 2]. For instance, a policy for a partially observable Markov decision process (POMDP) might be forced to disregard part of the available information in order to meet computational demands [3]. Cooperative multi-agent settings offer another such example: each agent might perceive only its surroundings and be unable to communicate with all other agents; hence, a policy specifying an agent's behavior must rely exclusively on local information [4]; it might be further constrained to a maximum size to be computationally tractable [5].

Influence diagrams [6] are representational devices for utility-based decision making under uncertainty. Many popular decision-making frameworks such as finite-horizon POMDPs can be casted as influence diagrams [7]. Traditionally, influence diagrams target problems involving a single, non-forgetful decision maker; this makes them unfitted to represent decision-making with limited information. Limited memory influence diagrams (LIMIDs) generalize influence diagrams to allow for (explicit representation of) bounded memory policies and simultaneous decisions [1, 2]. More precisely, LIMIDs relax the *regularity* and *no forgetting* assumptions of influence diagrams, namely, that there is a complete temporal ordering over the decisions, and that observations and decisions are permanently remembered.

Solving a LIMID refers to finding a combination of policies that maximizes expected utility. This task has been empirically and theoretically shown to be a very hard problem [8]. Under certain graph-structural conditions (which no forgetting and regularity imply), Lauritzen and Nilsson [2] show that LIMIDs can be solved by dynamic programming with complexity exponential in the treewidth of the graph. However, when these conditions are not met, their iterative algorithm might converge to a local optimum that is far from the optimum. Recently, de Campos and Ji [8] formulated the CR (Credal Reformulation) algorithm that solves a LIMID by mapping it into a mixed integer programming problem; they show that CR is able to solve small problems exactly and obtain good approximations for medium-sized problems.

In this paper, we formally describe LIMIDs (Section 2) and show that policies can be partially ordered, and that the ordering can be extended monotonically, allowing for the generalized variable elimination procedure in Section 3. We show experimentally in Section 4 that the algorithm built on these ideas can enormously save computational resources, allowing many problems to be solved exactly. In fact, our algorithm is orders of magnitude faster than the CR algorithm on randomly generated diagrams containing up to 150 variables. Finally, we write our conclusions in Section 5.

## 2 Limited memory influence diagrams

In the LIMID formalism, the quantities and events of interest are represented by three distinct types of variables or nodes: *chance variables* (oval nodes) represent events on which the decision maker has no control, such as outcomes of tests or consequences of actions; *decision variables* (square nodes) represent the alternatives a decision maker might have; *value variables* (diamond-shaped nodes) represent additive parcels of the overall utility. Let $\mathcal{U}$ be the set of all variables relevant to a problem. Each variable $X$ in $\mathcal{U}$ has associated a *domain* $\Omega_X$, which is the finite non-empty set of *values* or *states* $X$ can assume. The *empty domain* $\Omega_\emptyset \triangleq \{\lambda\}$ contains a single element $\lambda$ that is not in any other domain. Decision and chance variables have domains different from the empty domain, whereas value variables are always associated to the empty domain. The domain $\Omega_x$ of a set of variables $x = \{X_1, \ldots, X_n\} \subseteq \mathcal{U}$ is the Cartesian product $\Omega_{X_1} \times \cdots \times \Omega_{X_n}$ of the variable domains. If $x$ and $y$ are sets of variables such that $y \subseteq x \subseteq \mathcal{U}$, and $\boldsymbol{x}$ is an element of the domain $\Omega_x$, we write $\boldsymbol{x}^{\downarrow y}$ to denote the projection of $\boldsymbol{x}$ onto the smaller domain $\Omega_y$, that is, $\boldsymbol{x}^{\downarrow y} \in \Omega_y$ contains only the components of $\boldsymbol{x}$ that are compatible with the variables in $y$. By convention, $\boldsymbol{x}^{\downarrow \emptyset} \triangleq \lambda$. The cylindrical extension of $\boldsymbol{y} \in \Omega_y$ to $\Omega_x$ is the set $\boldsymbol{y}^{\uparrow x} \triangleq \{\boldsymbol{x} \in \Omega_x : \boldsymbol{x}^{\downarrow y} = \boldsymbol{y}\}$. Oftentimes, if clear from the context, we write $X_1 \cdots X_n$ to denote the set $\{X_1, \ldots, X_n\}$, and $X$ to denote $\{X\}$.

We notate point-wise comparison of functions implicitly. For example, if $f$ and $g$ are real-valued functions over a domain $\Omega_x$ and $k$ is a real number, we write $f \geq g$ and $f = k$ meaning $f(\boldsymbol{x}) \geq g(\boldsymbol{x})$ and $f(\boldsymbol{x}) = k$, respectively, for all $\boldsymbol{x} \in \Omega_x$. Any function over a domain containing a single element is identified by the real number it returns. If $f$ and $g$ are functions over domains $\Omega_x$ and $\Omega_y$, respectively, their product $fg$ is the function over $\Omega_{x \cup y}$ such that $(fg)(\boldsymbol{w}) = f(\boldsymbol{w}^{\downarrow x}) g(\boldsymbol{w}^{\downarrow y})$ for all $\boldsymbol{w}$. Sum of functions is defined analogously: $(f + g)(\boldsymbol{w}) = f(\boldsymbol{w}^{\downarrow x}) + g(\boldsymbol{w}^{\downarrow y})$. If $f$ is a function over $\Omega_x$, and $y \subseteq \mathcal{U}$, the *sum-marginal* $\sum_y f$ returns a function over $\Omega_{x \setminus y}$ such that for any element $\boldsymbol{w}$ of its domain we have $(\sum_y f)(\boldsymbol{w}) = \sum_{\boldsymbol{x} \in \boldsymbol{w}^{\uparrow x}} f(\boldsymbol{x})$. Notice that if $y \cap x = \emptyset$, then $\sum_y f = f$.

Let $\mathcal{C}$, $\mathcal{D}$ and $\mathcal{V}$ denote the sets of chance, decision and value variables, respectively, in $\mathcal{U}$. A LIMID $\mathcal{L}$ is an annotated direct acyclic graph (DAG) over the set of variables $\mathcal{U}$, where the nodes in $\mathcal{V}$ have no children. The precise meanings of the arcs in $\mathcal{L}$ vary according to the type of node to which they point. Arcs entering chance and value nodes denote stochastic and functional dependency, respectively; arcs entering decision nodes describe information awareness or relevance at the time the decision is made. If $X$ is a node in $\mathcal{L}$, we denote by $\mathtt{pa}_X$ the set of parents of $X$, that is, the set of nodes of $\mathcal{L}$ from which there is an arc pointing to $X$. Similarly, we let $\mathtt{ch}_X$ denote the set of children of $X$ (i.e., nodes to which there is an arc from $X$), and $\mathtt{fa}_X \triangleq \mathtt{pa}_X \cup \{X\}$ denote its family. Each chance variable $C$ in $\mathcal{C}$ has an associated function $p_C^{\mathtt{pa}_C}$ specifying the probability $\Pr(C = \boldsymbol{x}^{\downarrow C} | \mathtt{pa}_C = \boldsymbol{x}^{\downarrow \mathtt{pa}_C})$ of $C$ assuming value $\boldsymbol{x}^{\downarrow C} \in \Omega_C$ given that the parents take on values $\boldsymbol{x}^{\downarrow \mathtt{pa}_C} \in \Omega_{\mathtt{pa}_C}$ for all $\boldsymbol{x} \in \Omega_{\mathtt{fa}_C}$. We assume that the probabilities associated to any chance node respect the Markov condition, that is, that any variable $X \in \mathcal{C}$ is stochastically independent from its non-descendant non-parents given its parents. Each value variable $V \in \mathcal{V}$ is associated to a bounded real-valued utility function $u_V$ over $\Omega_{\mathtt{pa}_V}$, which quantifies the (additive) contribution of the states of its parents to the overall utility. Thus, the overall utility of a joint state $\boldsymbol{x} \in \Omega_{\mathcal{C} \cup \mathcal{D}}$ is given by the sum of utility functions $\sum_{V \in \mathcal{V}} u_V(\boldsymbol{x}^{\downarrow \mathtt{pa}_V})$. For any decision variable $D \in \mathcal{D}$, a *policy* $\delta_D$ specifies an action for each possible state configuration of its parents, that is, $\delta_D : \Omega_{\mathtt{pa}_D} \to \Omega_D$. If $D$ has no parents, then $\delta_D$ is a function from the empty domain to $\Omega_D$, and therefore constitutes a choice of $\boldsymbol{x} \in \Omega_D$. The set of all policies $\delta_D$ for a variable $D$ is denoted by $\Delta_D$.

To illustrate the use of LIMIDs, consider the following example involving a memoryless robot in a 5-by-5 gridworld (Figure 1a). The robot has 9 time steps to first reach a position $s_A$ of the grid, for which it receives 10 points, and then a position $s_B$, for which it is rewarded with 20 points. If the positions are visited in the wrong order, or if a point is re-visited, no reward is given. At each step, the robot can perform actions move north, south, east or west, which cost 1 point and succeed with 0.9 probability, or do nothing, which incurs no cost and always succeeds. Finally, the robot can estimate its position in the grid by measuring the distance to each of the four walls. The estimated position is correct 70% of the time, wrong by one square 20% of the time, and by two squares 10% of the time. The LIMID in Figure 1b formally represents the environment and the robot behavior.

The action taken by the robot at time step $t$ is represented by variable $D_t$ $(t = 1, \ldots, 8)$. The costs associated to decisions are represented by variables $C_t$, which have associated functions $u_{C_t}$ that

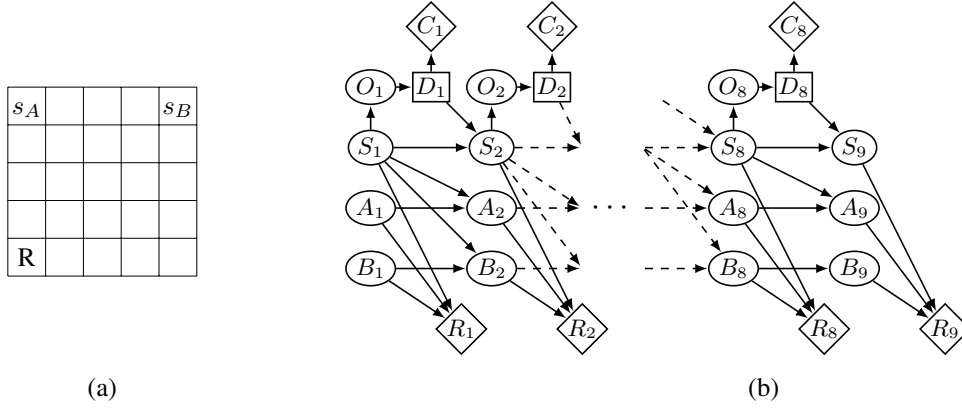

| | | | | | |
| $s_A$ | | | $s_B$ | |

| R | | | | |

(a)

(b)

Figure 1: (a) A robot R in a 5-by-5 gridworld with two goal-states. (b) The corresponding LIMID.

return zero if $D_t$ = nothing, and otherwise return -1. The variables $S_t$ ($t = 1, \ldots, 9$) represent the robot's actual position at time step $t$, while variables $O_t$ denote its estimated position. The function $p_{S_t}^{S_{t-1}D_t}$ associated to $S_t$ specifies the probabilities $\Pr(S_t = s_t | S_{t-1} = s_{t-1}, D_t = d_t)$ of transitioning to state $S_t = s_t$ from a state $S_{t-1} = s_{t-1}$ when the robot executes action $D_t = d_t$. The function $p_{O_t}^{S_t}$ is associated to $O_t$ and quantifies the likelihood of estimating position $O_t = o_t$ when in position $S_t = s_t$. We use binary variables $A_t$ and $B_t$ to denote whether positions $s_A$ and $s_B$, respectively, have been visited by the robot before time step $t$. Hence, the function $p_{A_t}^{A_{t-1}S_{t-1}}$ associated to $A_t$ equals one for $A_t = y$ if $S_{t-1} = s_a$ or $A_{t-1} = y$, and zero otherwise. Likewise, the function $p_{B_t}^{B_{t-1}S_{t-1}}$ equals one for $B_t = y$ only if either $S_{t-1} = s_B$ or $B_{t-1} = y$. The reward received by the robot in step $t$ is represented by variable $R_t$. The utility function $u_{R_t}$ associated to $R_t$ equals 10 if $s_t = s_A$ and $A_t = n$ and $B_t = n$, 20 if $s_t = s_B$ and $A_t = y$ and $B_t = n$, and zero otherwise.

Let $\Delta \triangleq \times_{D \in \mathcal{D}} \Delta_D$ denote the space of possible combinations of policies. An element $s = (\delta_D)_{D \in \mathcal{D}} \in \Delta$ is said to be a *strategy* for $\mathcal{L}$. Given a policy $\delta_D$, let $p_D^{\mathtt{pa}_D}$ denote a function such that for each $\boldsymbol{x} \in \Omega_{\mathtt{fa}_D}$ it equals one if $\boldsymbol{x}^{\downarrow D} = \delta_D(\boldsymbol{x}^{\downarrow \mathtt{pa}_D})$ and zero otherwise. In other words, $p_D^{\mathtt{pa}_D}$ is a conditional probability table representing policy $\delta_D$. There is a one-to-one correspondence between functions $p_D^{\mathtt{pa}_D}$ and policies $\delta_D \in \Delta_D$, and specifying a policy $\delta_D$ is equivalent to specifying $p_D^{\mathtt{pa}_D}$. We denote the set of all functions $p_D^{\mathtt{pa}_D}$ by $\mathcal{P}_D$. A strategy $s$ induces a joint probability mass function over the variables in $\mathcal{C} \cup \mathcal{D}$ by

$$p_s \triangleq \prod_{C \in \mathcal{C}} p_C^{\mathtt{pa}_C} \prod_{D \in \mathcal{D}} p_D^{\mathtt{pa}_D}, \tag{1}$$

and has an associated expected utility given by

$$\mathrm{E}_s[\mathcal{L}] \triangleq \sum_{\boldsymbol{x} \in \Omega_{\mathcal{C} \cup \mathcal{D}}} p_s(\boldsymbol{x}) \sum_{V \in \mathcal{V}} u_V(\boldsymbol{x}^{\downarrow \mathtt{pa}_V}) = \sum_{\mathcal{C} \cup \mathcal{D}} p_s \sum_{V \in \mathcal{V}} u_V. \tag{2}$$

The *treewidth* of a graph measures its resemblance to a tree and is given by the number of vertices in the largest clique of the corresponding triangulated moral graph minus one. Given a LIMID $\mathcal{L}$ of treewidth $\omega$, we can evaluate the expected utility of any strategy $s$ in time and space at most exponential in $\omega$. Hence, if $\omega$ is bounded by a constant, computing $\mathrm{E}_s[\mathcal{L}]$ takes polynomial time [9]. The primary task of a LIMID is to find an optimal strategy $s^*$ with maximal expected utility, that is, to find $s^*$ such that $\mathrm{E}_s[\mathcal{L}] \leq \mathrm{E}_{s^*}[\mathcal{L}]$ for all $s \in \Delta$. The value $\mathrm{E}_{s^*}[\mathcal{L}]$ is called the *maximum expected utility* of $\mathcal{L}$ and it is denoted by $\mathrm{MEU}[\mathcal{L}]$. In the LIMID of Figure 1, the goal is to find an optimal strategy $s = (\delta_{D_1}, \ldots, \delta_{D_8})$, where the optimal policies $\delta_{D_t}$ for $t = 1, \ldots, 8$ prescribe an action in $\Omega_{D_t} = \{\text{north, south, west, east, nothing}\}$ for each possible estimated position in $\Omega_{O_t}$.

For most real problems, enumerating all the strategies is prohibitively costly. In fact, computing the MEU is NP-hard even in bounded treewidth diagrams [8]. It is well-known that any LIMID $\mathcal{L}$ can be mapped into an equivalent LIMID $\mathcal{L}'$ where all utilities take values on the real interval $[0, 1]$ [10]. The mapping preserves optimality of strategies, that is, any optimal strategy for $\mathcal{L}'$ is also an optimal

strategy for $\mathcal{L}$ (and vice-versa). This allows us, in the rest of the paper, to focus on LIMIDs whose utilities are defined in $[0, 1]$ with no loss of generality for the algorithm we devise.

## 3 A fast algorithm for solving LIMIDs exactly

The basic ingredients of our algorithmic framework for representing and handling information in LIMIDs are the so-called *valuations*, which encode information (probabilities, utilities and policies) about the elements of a domain. Each valuation is associated to a subset of the variables in $\mathcal{U}$, called its *scope*. More concretely, we define a valuation $\phi$ with scope $x$ as a pair $(p, u)$ of bounded nonnegative real-valued functions $p$ and $u$ over the domain $\Omega_x$; we refer to $p$ and $u$ as the probability and utility part, respectively, of $\phi$. Often, we write $\phi_x$ to make explicit the scope $x$ of a valuation $\phi$. For any $x \subseteq \mathcal{U}$, we denote the set of all possible valuations with scope $x$ by $\Phi_x$. The set of all possible valuations is given by $\Phi \triangleq \bigcup_{x \subseteq \mathcal{U}} \Phi_x$. The set $\Phi$ is closed under the operations of *combination* and *marginalization*. Combination represents the aggregation of information and is defined as follows. If $\phi = (p, u)$ and $\psi = (q, v)$ are valuations with scopes $x$ and $y$, respectively, its combination $\phi \otimes \psi$ is the valuation $(pq, pv + qu)$ with scope $x \cup y$. Marginalization, on the other hand, acts by coarsening information. If $\phi = (p, u)$ is a valuation with scope $x$, and $y$ is a set of variables such that $y \subseteq x$, the marginal $\phi^{\downarrow y}$ is the valuation $(\sum_{x \setminus y} p, \sum_{x \setminus y} u)$ with scope $y$. In this case, we say that $z \triangleq x \setminus y$ has been *eliminated* from $\phi$, which we denote by $\phi^{-z}$. The following result shows that our framework respects the necessary conditions for computing efficiently with valuations (in the sense of keeping the scope of valuations minimal during the variable elimination procedure).

**Proposition 1.** *The system* $(\Phi, \mathcal{U}, \otimes, \downarrow)$ *satisfies the following three axioms of a (weak)* labeled valuation algebra *[11, 12].*

*(A1) For any $\phi_1, \phi_2, \phi_3 \in \Phi$ we have that $\phi_1 \otimes \phi_2 = \phi_2 \otimes \phi_1$ and $\phi_1 \otimes (\phi_2 \otimes \phi_3) = (\phi_1 \otimes \phi_2) \otimes \phi_3$.*

*(A2) For any $\phi_z \in \Phi_z$ and $y \subseteq x \subseteq z$ we have that $(\phi_z^{\downarrow x})^{\downarrow y} = \phi_z^{\downarrow y}$.*

*(A3) For any $\phi_x \in \Phi_x$, $\phi_y \in \Phi_y$ and $x \subseteq z \subseteq x \cup y$ we have that $(\phi_x \otimes \phi_y)^{\downarrow z} = \phi_x \otimes \phi_y^{\downarrow y \cap z}$.*

*Proof.* (A1) follows directly from commutativity, associativity and distributivity of product and sum of real-valued functions, and (A2) follows directly from commutativity of the sum-marginal operation. To show (A3), consider any two valuations $(p, u)$ and $(q, v)$ with scopes $x$ and $y$, respectively, and a set $z$ such that $x \subseteq z \subseteq x \cup y$. By definition of combination and marginalization, we have that $[(p, u) \otimes (q, v)]^{\downarrow z} = (\sum_{x \cup y \setminus z} pq, \sum_{x \cup y \setminus z} (pv + qu))$. Since $x \cup y \setminus z = y \setminus z$, and $p$ and $u$ are functions over $\Omega_x$, it follows that $(\sum_{x \cup y \setminus z} pq, \sum_{x \cup y \setminus z} (pv + qu)) = (p \sum_{y \setminus z} q, p \sum_{y \setminus z} v + u \sum_{y \setminus z} q)$, which equals $(p, u) \otimes (\sum_{y \setminus z} q, \sum_{y \setminus z} v) = (p, y) \otimes (q, v)^{\downarrow y \cap z}$. Hence, $[(p, u) \otimes (q, v)]^{\downarrow z} = (p, y) \otimes (q, v)^{\downarrow y \cap z}$. $\qquad\square$

The following lemma is a direct consequence of (A3) shown by [12], required to prove the correctness of our algorithm later on.

**Lemma 2.** *If $z \subseteq y$ and $z \cap x = \emptyset$ then $(\phi_x \otimes \phi_y)^{-z} = \phi_x \otimes \phi_y^{-z}$.*

The framework of valuations allows us to compute the expected utility of a given strategy efficiently:

**Proposition 3.** *Given a LIMID $\mathcal{L}$ and a strategy $s = (\delta_D)_{D \in \mathcal{D}}$, let*

$$\phi_s \triangleq \left[ \bigotimes_{C \in \mathcal{C}} (p_C^{\mathtt{pa}_C}, 0) \right] \otimes \left[ \bigotimes_{D \in \mathcal{D}} (p_D^{\mathtt{pa}_D}, 0) \right] \otimes \left[ \bigotimes_{V \in \mathcal{V}} (1, u_V) \right], \tag{3}$$

*where, for each $D$, $p_D^{\mathtt{pa}_D}$ is the function in $\mathcal{P}_D$ associated with policy $\delta_D$. Then $\phi_s^{\downarrow \emptyset} = (1, \mathrm{E}_s[\mathcal{L}])$.*

*Proof.* Let $p$ and $u$ denote the probability and utility part, respectively, of $\phi_s^{\downarrow \emptyset}$. By definition of combination, we have that $\phi_s = (p_s, p_s \sum_{V \in \mathcal{V}} u_V)$, where $p_s = \prod_{X \in \mathcal{C} \cup \mathcal{D}} p_X^{\mathtt{pa}_X}$ as in (1). Since $p_s$ is a probability distribution over $\mathcal{C} \cup \mathcal{D}$, it follows that $p = \sum_{\boldsymbol{x} \in \Omega_{\mathcal{C} \cup \mathcal{D}}} p_s(\boldsymbol{x}) = 1$. Finally, $u = \sum_{\mathcal{C} \cup \mathcal{D}} p_s \sum_{V \in \mathcal{V}} u_V$, which equals $\mathrm{E}_s[\mathcal{L}]$ by (2). $\qquad\square$

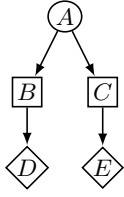

**Input**: elimination ordering $B < C < A$ and strategy $s = (\delta_B, \delta_C)$
**Initialization**:

$$\phi_A = (p_A, 0) \quad \phi_B = (p_B^A, 0) \quad \phi_C = (p_C^A, 0) \quad \phi_D = (1, u_D) \quad \phi_E = (1, u_E)$$

**Propagation**:

$$\psi_1 = (\phi_B \otimes \phi_D)^{-B} \qquad \psi_2 = (\phi_C \otimes \phi_E)^{-C} \qquad \psi_3 = (\psi_1 \otimes \psi_2 \otimes \phi_A)^{-A}$$

**Termination**: return the utility part of $\phi_s^{\downarrow\emptyset} = \psi_3$

Figure 2: Computing the expected utility of a strategy by variable elimination.

Given any strategy $s$, we can use a variable elimination procedure to efficiently compute $\phi_s^{\downarrow\emptyset}$ and hence its expected utility in time polynomial in the largest domain of a variable but exponential in the width of the elimination ordering.[1] Figure 2 shows a variable elimination procedure used to compute the expected utility of a strategy of the simple LIMID on the left-hand side. However, computing the MEU in this way is unfeasible for any reasonable diagram due to the large number of strategies that would need to be enumerated. For example, if the variables $A$, $B$ and $C$ in the LIMID in Figure 2 have each ten states, there are $10^{10}10^{10} = 10^{20}$ possible strategies.

In order to avoid considering all possible strategies, we define a partial order (i.e., a reflexive, antisymmetric and transitive relation) over $\Phi$ as follows. For any two valuations $\phi = (p, u)$ and $\psi = (q, v)$ in $\Phi$, if $\phi$ and $\psi$ have equal scope, $p \le q$ and $u \le v$, then $\phi \le \psi$ holds. The following result shows that $\le$ is monotonic with respect to combination and marginalization.

**Proposition 4.** *The system* $(\Phi, \mathcal{U}, \otimes, \downarrow, \le)$ *satisfies the following two additional axioms of an* ordered valuation algebra *[13]*.

*(A4) If $\phi_x \le \psi_x$ and $\phi_y \le \psi_y$, then $(\phi_x \otimes \phi_y) \le (\psi_x \otimes \psi_y)$.*

*(A5) If $\phi_x \le \psi_x$ then $\phi_x^{\downarrow y} \le \psi_x^{\downarrow y}$.*

*Proof.* (A4). Consider two valuations $(p_x, u_x)$ and $(q_x, v_x)$ with scope $x$ such that $(p_x, u_x) \le (q_x, v_x)$, and two valuations $(p_y, u_y)$ and $(q_y, v_y)$ with scope $y$ satisfying $(p_y, u_y) \le (q_y, v_y)$. By definition of $\le$, we have that $p_x \le q_x$, $u_x \le v_x$, $p_y \le q_y$ and $u_y \le v_y$. Since all functions are nonnegative, it follows that $p_x p_y \le q_x q_y$, $p_x u_y \le q_x v_y$ and $p_y u_x \le q_y v_x$. Hence, $(p_x, u_x) \otimes (p_y, u_y) = (p_x p_y, p_x u_y + p_y u_x) \le (q_x q_y, q_x v_y + q_y v_x) = (q_x, v_x) \otimes (q_y, v_y)$. (A5). Let $y$ be a subset of $x$. It follows from monotonicity of $\le$ with respect to addition of real numbers that $(p_x, u_x)^{\downarrow y} = (\sum_{x \backslash y} p_x, \sum_{x \backslash y} u_x) \le (\sum_{x \backslash y} q_x, \sum_{x \backslash y} v_x) = (q_x, v_x)^{\downarrow y}$. $\square$

The monotonicity of $\le$ allows us to detect suboptimal strategies during variable elimination. To illustrate this, consider the variable elimination scheme in Figure 2 for two different strategies $s$ and $s'$, and let $\psi_1^s, \psi_2^s, \psi_3^s$ be the valuations produced in the propagation step for strategy $s$ and $\psi_1^{s'}, \psi_2^{s'}, \psi_3^{s'}$ the valuations for $s'$. If $\psi_1^s \le \psi_1^{s'}$ and $\psi_2^s \le \psi_2^{s'}$ then Proposition 4 tells us that $\psi_3^s \le \psi_3^{s'}$, which implies $E_s[\mathcal{L}] \le E_{s'}[\mathcal{L}]$. As a consequence, we can abort variable elimination for $s$ after the second iteration. We can also exploit the redundancy between valuations produced during variable elimination for neighbor strategies. For example, if $s$ and $s'$ specify the same policy for $B$, then we know in advance that $\psi_1^s = \psi_2^{s'}$, so that only one of them needs to be computed.

In order to facilitate the description of our algorithm, we define operations over sets of valuations. If $\Psi_x$ is a set of valuations with scope $x$ and $\Psi_y$ is a set of valuations with scope $y$ the operation $\Psi_x \otimes \Psi_y \triangleq \{\phi_x \otimes \phi_y : \phi_x \in \Psi_x, \phi_y \in \Psi_y\}$ returns the set of combinations of a valuation in $\Psi_x$ and a valuation in $\Psi_y$. For $X \in x$, the operation $\Psi_x^{-X} \triangleq \{\phi_x^{-X} : \phi_x \in \Psi_x\}$ eliminates variable $X$ from all valuations in $\Psi_x$. Given a finite set of valuations $\Psi \subseteq \Phi$, we say that a valuation $\phi \in \Psi$ is *maximal* if for all $\psi \in \Psi$ such that $\phi \le \psi$ it holds that $\psi \le \phi$. The operator prune returns the set prune$(\Psi)$ of maximal valuations of $\Psi$ (by pruning non-maximal valuations).

We are now ready to describe the **Multiple Policy Updating** (MPU) algorithm, which solves arbitrary LIMIDs exactly. Consider a LIMID $\mathcal{L}$ and an elimination ordering $X_1 < \cdots < X_n$ over the variables in $\mathcal{C} \cup \mathcal{D}$. The elimination ordering can be selected using the standard methods for Bayesian networks [9]. Note that unlike standard algorithms for variable elimination in influence diagrams we allow any elimination ordering. The algorithm is initialized by generating one set of valuations for each variable $X$ in $\mathcal{U}$ as follows.

**Initialization:** Let $\mathcal{V}_0$ be initially the empty set.

1. For each chance variable $X \in \mathcal{C}$, add a singleton $\Psi_X \triangleq \{(p_X^{\mathtt{pa}_X}, 0)\}$ to $\mathcal{V}_0$;

2. For each decision variable $X \in \mathcal{D}$, add a set of valuations $\Psi_X \triangleq \{(p_X^{\mathtt{pa}_X}, 0) : p_X^{\mathtt{pa}_X} \in \mathcal{P}_X\}$ to $\mathcal{V}_0$;

3. For each value variable $X \in \mathcal{V}$, add a singleton $\Psi_X \triangleq \{(1, u_X)\}$ to $\mathcal{V}_0$.

Once $\mathcal{V}_0$ has been initialized with a set of valuations for each variable in the diagram, we recursively eliminate a variable $X_i$ in $\mathcal{C} \cup \mathcal{D}$ in the given ordering and remove any non-maximal valuation:

**Propagation:** For $i = 1, \ldots, n$ do:

1. Let $\mathcal{B}_i$ be the set of all valuations in $\mathcal{V}_{i-1}$ whose scope contains $X_i$;

2. Compute $\Psi_i \triangleq \mathrm{prune}([\bigotimes_{\Psi \in \mathcal{B}_i} \Psi]^{-X_i})$;

3. Set $\mathcal{V}_i \triangleq \mathcal{V}_{i-1} \cup \{\Psi_i\} \setminus \mathcal{B}_i$.

Finally, the algorithm outputs the utility part of the single maximal valuation in the set $\bigotimes_{\Psi \in \mathcal{V}_n} \Psi$:

**Termination:** Return the real number $u$ such that $(p, u) \in \mathrm{prune}(\bigotimes_{\Psi \in \mathcal{V}_n} \Psi)$.

$u$ is a real number because the valuations in $\bigotimes_{\Psi \in \mathcal{V}_n} \Psi$ have empty scope and thus both their probability and utility parts are identified with real numbers. The following result is a straightforward extension of [14, Lemma 1(iv)] that is needed to guarantee the correctness of discarding non-maximal valuations in the propagation step.

**Lemma 5.** *(Distributivity of maximality). If $\Psi_x$ and $\Psi_y$ are two sets of ordered valuations and $z \subseteq x$ then (i)* $\mathrm{prune}(\Psi_x \otimes \mathrm{prune}(\Psi_y)) = \mathrm{prune}(\Psi_x \otimes \Psi_y)$ *and (ii)* $\mathrm{prune}(\mathrm{prune}(\Psi_x)^{\downarrow z}) = \mathrm{prune}(\Psi_x^{\downarrow z})$.

The result shows that, like marginalization, the $\mathrm{prune}$ operation distributes over any factorization of $\bigotimes_{X \in \mathcal{U}} \Psi_X$. The following lemma shows that at any iteration $i$ of the propagation step the combination of all sets in the current pool of sets $\mathcal{V}_i$ produces the set of maximal valuations of the initial factorization.

**Lemma 6.** *For $i \in \{1, \ldots, n\}$, it follows that* $\mathrm{prune}([\bigotimes_{\Psi \in \mathcal{V}_0} \Psi]^{-X_1 \cdots X_i}) = \mathrm{prune}(\bigotimes_{\Psi \in \mathcal{V}_i} \Psi)$.

*Proof.* We show the result by induction on $i$. The basis is easily obtained by applying Lemmas 2 and 5 and the axioms of valuation algebra to $\mathrm{prune}([\bigotimes_{\Psi \in \mathcal{V}_0} \Psi]^{-X_1})$ in order to obtain $\mathrm{prune}(\bigotimes_{\Psi \in \mathcal{V}_1} \Psi)$. For the induction step, assume the result holds at $i$, that is, $\mathrm{prune}([\bigotimes_{\Psi \in \mathcal{V}_0} \Psi]^{-X_1 \cdots X_i}) = \mathrm{prune}(\bigotimes_{\Psi \in \mathcal{V}_i} \Psi)$. By eliminating $X_{i+1}$ from both sides and then applying the prune operation we get to $\mathrm{prune}([\mathrm{prune}([\bigotimes_{\Psi \in \mathcal{V}_0} \Psi]^{-X_1 \cdots X_i})]^{-X_{i+1}}) = \mathrm{prune}([\mathrm{prune}(\bigotimes_{\Psi \in \mathcal{V}_i} \Psi)]^{-X_{i+1}})$. By Lemma 5(ii) and (A2), we have that $\mathrm{prune}([\bigotimes_{\Psi \in \mathcal{V}_0} \Psi]^{-\{X_1 \cdots X_{i+1}\}}) = \mathrm{prune}([\bigotimes_{\Psi \in \mathcal{V}_i} \Psi]^{-X_{i+1}})$. It follows from (A1) and Lemma 2 that the right-hand part equals $\mathrm{prune}((\bigotimes_{\Psi \in \mathcal{V}_i \setminus \mathcal{B}_{i+1}} \Psi) \otimes [(\bigotimes_{\Psi \in \mathcal{B}_{i+1}} \Psi)]^{-X_{i+1}})$, which by Lemma 5(i) equals $\mathrm{prune}((\bigotimes_{\Psi \in \mathcal{V}_i \setminus \mathcal{B}_{i+1}} \Psi) \otimes \mathrm{prune}([(\bigotimes_{\Psi \in \mathcal{B}_{i+1}} \Psi)]^{-X_{i+1}}))$, which by definition of $\mathcal{V}_{i+1}$ equals $\mathrm{prune}(\bigotimes_{\Psi \in \mathcal{V}_{i+1}} \Psi)$. $\square$

Let $\Psi_{\mathcal{L}} \triangleq \{\phi_s : s \in \Delta\}$, where $\phi_s$ is given by (3). According to Proposition 3, each element $\phi_s^{-X_1 \cdots X_n}$ in $\Psi_{\mathcal{L}}^{-X_1 \cdots X_n}$ is a valuation whose probability part is one and utility part equals $\mathrm{E}_s[\mathcal{L}]$. Thus, the maximal expected utility $\mathrm{MEU}[\mathcal{L}]$ is the utility part of the single valuation in $\mathrm{prune}(\Psi_{\mathcal{L}}^{-X_1 \cdots X_n})$. It is not difficult to see that after the initialization step, the set $\mathcal{V}_0$ contains sets

$\Psi$ of valuations such that $\bigotimes_{\Psi \in \mathcal{V}_0} \Psi = \Psi_{\mathcal{L}}$. Hence, Lemma 6 states that after the last iteration, MPU produces a set $\mathcal{V}_n$ of sets of valuations such that $\text{prune}(\bigotimes_{\Psi \in \mathcal{V}_n} \Psi) = \text{prune}(\Psi_{\mathcal{L}}^{-X_1 \cdots X_n}) = \text{MEU}[\mathcal{L}]$. This is precisely what the following theorem shows.

**Theorem 7.** *Given a LIMID $\mathcal{L}$, MPU outputs* $\text{MEU}[\mathcal{L}]$.

*Proof.* The algorithm returns the utility part of a valuation $(p, u)$ in $\text{prune}(\bigotimes_{\Psi \in \mathcal{V}_n} \Psi)$, which, by Lemma 6 for $i = n$, equals $\text{prune}([\bigotimes_{\Psi \in \mathcal{V}_0} \Psi]^{\downarrow \emptyset})$. By definition of $\mathcal{V}_0$, any valuation $\phi$ in $(\bigotimes_{\Psi \in \mathcal{V}_0} \Psi)$ factorizes as in (3). Also, there is exactly one valuation $\phi \in (\bigotimes_{\Psi \in \mathcal{V}_0} \Psi)$ for each strategy in $\Delta$. Hence, by Proposition 3, the set $(\bigotimes_{\Psi \in \mathcal{V}_0} \Psi)^{\downarrow \emptyset}$ contains a pair $(1, \text{E}_s[\mathcal{L}])$ for every strategy $s$ inducing a distinct expected utility. Moreover, since functions with empty scope correspond to numbers, the relation $\leq$ specifies a total ordering over the valuations in $(\bigotimes_{\Psi \in \mathcal{V}_0} \Psi)^{\downarrow \emptyset}$, which implies a single maximal element. Let $s^*$ be a strategy associated to $(p, u)$. Since $(p, u) \in \text{prune}([\bigotimes_{\Psi \in \mathcal{V}_0} \Psi]^{\downarrow \emptyset})$, it follows from maximality that $\text{E}_{s^*}[\mathcal{L}] \geq \text{E}_s[\mathcal{L}]$ for all $s$, and hence $u = \text{MEU}[\mathcal{L}]$. $\qquad \square$

The time complexity of the algorithm is given by the cost of creating the sets of valuations in the initialization step plus the overall cost of the combination and marginalization operations performed during the propagation step. Regarding the initialization step, the loops for chance and value variables generate singletons, and thus take time linear in the input. For any decision variable $D$, let $\rho_D \triangleq |\Omega_D|^{|\Omega_{\text{pa}_D}|}$ denote the number of policies in $\Delta_D$ (which coincides with the number of functions in $\mathcal{P}_D$). There is exactly one valuation in $\Psi_D$ in $\mathcal{V}_0$ for every policy in $\Delta_D$. Also, let $\rho \triangleq \text{prune}_{D \in \mathcal{D}} \rho_D$ be the cardinality of the largest policy set. Then the initialization loop for decision variables takes $O(|\mathcal{D}|\rho)$ time, which is exponential in the input (the sets of policies are not considered as an input of the problem). Let us analyze the propagation step. As with any variable elimination procedure, the running time of propagating (sets of) valuations is exponential in the width of the given ordering, which is in the best case given by the treewidth of the diagram. Consider the case of an ordering with bounded width $\omega$ and a bounded number of states per variable $\kappa$. Then the cost of each combination or marginalization is bounded by a constant, and the complexity depends only on the number of operations performed. Let $\nu$ denote the cardinality of the largest set $\Psi_i$, for $i = 1, \ldots, n$. Computing $\Psi_i$ requires at most $\nu^{|\mathcal{U}|-1}$ operations of combination and $\nu$ operations of marginalization. In the worst case, $\nu$ is equal to $\rho^{|\mathcal{D}|} \leq O(\kappa^{|D|\kappa^{\omega}})$, that is, all sets associated to decision variables have been combined without discarding any valuation. Hence, the worst-case complexity of the propagation step is exponential in the input, even if the ordering width and the number of states per variable are bounded. This is not surprising given that the problem is still NP-hard in these cases. However, this is a very pessimistic scenario and, on average, the removal of non-maximal elements greatly reduces the complexity, as we show in the next section.

## 4 Experiments

We evaluate the performance of the algorithms on random LIMIDs generated in the following way. Each LIMID is parameterized by the number of decision nodes $d$, the number of chance nodes $c$, the maximum cardinality of the domain of a chance variable $\omega_C$, and the maximum cardinality of the domain of a decision variable $\omega_D$. We set the number of value nodes $v$ to be $d + 2$. For each variable $X_i$, $i = 1, \ldots, c + d + v$, we sample $\Omega_{X_i}$ to contain from 2 to 4 states. Then we repeatedly add an arc from a decision node with no children to a value node with no parents (so that each decision node has at least one value node as children). This step guarantees that all decisions are relevant for the computation of the MEU. Finally, we repeatedly add an arc that neither makes the domain of a variable greater than the given bounds nor makes the treewidth more than 10, until no arcs can be added without exceeding the bounds.[2] Note that this generates diagrams where decision and chance variables have at most $\log_2 \omega_D - 1$ and $\log_2 \omega_C - 1$ parents, respectively. Once the graph structure is obtained, we specify the functions associated to value variable by randomly sampling numbers in $[0, 1]$. The probability mass functions associated to chance variables are randomly sampled from a uniform prior distribution.

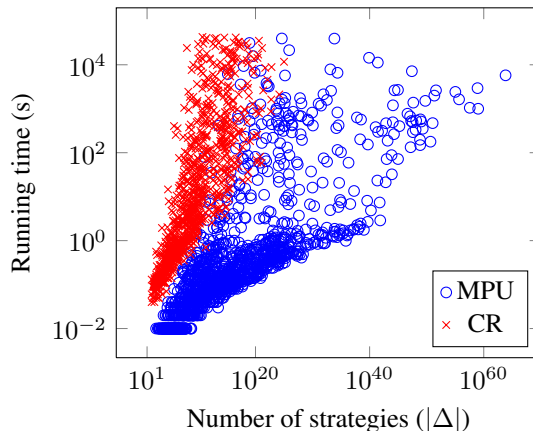

Figure 3: Running time of MPU and CR on randomly generated LIMIDs.

We compare MPU against the CR algorithm of [8] in 2530 LIMIDs randomly generated by the described procedure with parameters $5 \leq d \leq 50$, $8 \leq c \leq 50$, $8 \leq \omega_D \leq 64$ and $16 \leq \omega_C \leq 64$. MPU was implemented in C++ and tested in the same computer as CR.[3] A good succinct indicator of the hardness of solving a LIMID is the total number of strategies $|\Delta|$, which represents the size of the search space in a brute-force approach. $|\Delta|$ can also be loosely interpreted as the total number of alternatives (over all decision variables) in the problem instance. Figure 3 depicts running time against number of strategies in a log-log scale for the two algorithms on the same test set of random diagrams. For each algorithm, only solved instances are shown, which covers approximately $96\%$ of the cases for MPU, and $68\%$ for CR. A diagram is consider unsolved by an algorithm, if the algorithm was not able to reach the exact solution within the limit of 12 hours. Since CR uses an integer program solver, it can output a feasible solution within any given time limit; we consider a diagram solved by CR only if the solution returned at the end of 12 hours is exact, that is, only if its upper and lower bound values match. We note that MPU solved all cases that CR solved (but not the opposite). From the plot, one can see that MPU is orders of magnitude faster than CR. Within the limit of 12 hours, MPU was able to compute diagrams containing up to $10^{64}$ strategies, whereas CR solved diagrams with at most $10^{25}$ strategies. We remark that when CR was not able to solve a diagram, it almost always returned a solution that was not within 5% of the optimum. This implies that MPU would outperform CR even if the latter was allowed a small imprecision in its output.

## 5    Conclusion

LIMIDs are highly expressive models for utility-based decision making that subsume influence diagrams and finite-horizon (partially observable) Markov decision processes. Furthermore, they allow constraints on policies to be explicitly represented in a concise and intuitive graphical language. Unfortunately, solving LIMIDs is a very hard task of combinatorial optimization. Nevertheless, we showed here that our MPU algorithm can solve a large number of randomly generated problems in reasonable time. The algorithm efficiency is based on the early removal of suboptimal solutions, which drastically reduces the search space. An interesting extension is to improve MPU's running time at the expense of accuracy. This can be done by arbitrarily discarding valuations during the propagation step so as to bound the size of propagated sets. Future work is necessary to validate the feasibility of this idea.

**Acknowledgments**

This work was partially supported by the Swiss NSF grant nr. 200020_134759 / 1, and by the Computational Life Sciences Project, Canton Ticino.

## Footnotes

[1]The width of an elimination ordering is the maximum cardinality of the scope of a valuation produced during variable elimination minus one.

[2] Checking the treewidth of a graph might be hard. We instead use a greedy heuristic that resulted in diagrams whose treewidth ranged from 5 to 10.

[3]We used the CR implementation available at `http://www.idsia.ch/~cassio/id2mip/` and CPLEX [15] as mixed integer programming solver. Our MPU implementation can be downloaded at `http://www.idsia.ch/~cassio/mpu/`

## References

[1] N. L. Zhang, R. Qi, and D. Poole. A computational theory of decision networks. *International Journal of Approximate Reasoning*, 11(2):83–158, 1994.

[2] S. L. Lauritzen and D. Nilsson. Representing and solving decision problems with limited information. *Management Science*, 47:1235–1251, 2001.

[3] P. Poupart and C. Boutilier. Bounded finite state controllers. In *Advances in Neural Information Processing Systems 16 (NIPS)*, 2003.

[4] A. Detwarasiti and R. D. Shachter. Influence diagrams for team decision analysis. *Decision Analysis*, 2(4):207–228, 2005.

[5] C. Amato, D. S. Bernstein, and S. Zilberstein. Optimizing fixed-size stochastic controllers for POMDPs and decentralized POMDPs. *Autonomous Agents and Multi-Agent Systems*, 21(3):293–320, 2010.

[6] R. A. Howard and J. E. Matheson. Influence diagrams. In *Readings on the Principles and Applications of Decision Analysis*, pages 721–762. Strategic Decisions Group, 1984.

[7] J. A. Tatman and R. D. Shachter. Dynamic programming and influence diagrams. *IEEE Transactions on Systems, Man and Cybernetics*, 20(2):365–379, 1990.

[8] C. P. de Campos and Q. Ji. Strategy selection in influence diagrams using imprecise probabilities. In *Proceedings of the 24th Conference in Uncertainty in Artificial Intelligence*, pages 121–128, 2008.

[9] D. Koller and N. Friedman. *Probabilistic Graphical Models - Principles and Techniques*. MIT Press, 2009.

[10] G. F. Cooper. A method for using belief networks as influence diagrams. Fourth Workshop on Uncertainty in Artificial Intelligence, 1988.

[11] P. Shenoy and G. Shafer. Axioms for probability and belief-function propagation. In *Proceedings of the Fourth Conference on Uncertainty in Artificial Intelligence*, pages 169–198. Elsevier Science, 1988.

[12] J. Kohlas. *Information Algebras: Generic Structures for Inference*. Springer-Verlag, 2003.

[13] R. Haenni. Ordered valuation algebras: a generic framework for approximating inference. *International Journal of Approximate Reasoning*, 37(1):1–41, 2004.

[14] H. Fargier, E. Rollon, and N. Wilson. Enabling local computation for partially ordered preferences. *Constraints*, 15:516–539, 2010.

[15] Ilog Optimization. CPLEX documentation. http://www.ilog.com, 1990.

